# Training a Quantum Neural Network

**Bob Ricks**
Department of Computer Science
Brigham Young University
Provo, UT 84602
cyberbob@cs.byu.edu

**Dan Ventura**
Department of Computer Science
Brigham Young University
Provo, UT 84602
ventura@cs.byu.edu

## Abstract

Most proposals for quantum neural networks have skipped over the problem of how to train the networks. The mechanics of quantum computing are different enough from classical computing that the issue of training should be treated in detail. We propose a simple quantum neural network and a training method for it. It can be shown that this algorithm works in quantum systems. Results on several real-world data sets show that this algorithm can train the proposed quantum neural networks, and that it has some advantages over classical learning algorithms.

## 1   Introduction

Many quantum neural networks have been proposed [1], but very few of these proposals have attempted to provide an in-depth method of training them. Most either do not mention how the network will be trained or simply state that they use a standard gradient descent algorithm. This assumes that training a quantum neural network will be straightforward and analogous to classical methods. While some quantum neural networks seem quite similar to classical networks [2], others have proposed quantum networks that are vastly different [3, 4, 5]. Several different network structures have been proposed, including lattices [6] and dots [4]. Several of these networks also employ methods which are speculative or difficult to do in quantum systems [7, 8]. These significant differences between classical networks and quantum neural networks, as well as the problems associated with quantum computation itself, require us to look more deeply at the issue of training quantum neural networks. Furthermore, no one has done empirical testing on their training methods to show that their methods work with real-world problems.

It is an open question what advantages a quantum neural network (QNN) would have over a classical network. It has been shown that QNNs should have roughly the same computational power as classical networks [7]. Other results have shown that QNNs may work best with some classical components as well as quantum components [2].

Quantum searches can be proven to be faster than comparable classical searches. We leverage this idea to propose a new training method for a simple QNN. This paper details such a network and how training could be done on it. Results from testing the algorithm on several real-world problems show that it works.

# 2 Quantum Computation

Several necessary ideas that form the basis for the study of quantum computation are briefly reviewed here. For a good treatment of the subject, see [9].

## 2.1 Linear Superposition

*Linear superposition* is closely related to the familiar mathematical principle of linear combination of vectors. Quantum systems are described by a wave function $\psi$ that exists in a Hilbert space. The Hilbert space has a set of states, $|\phi_i\rangle$, that form a basis, and the system is described by a quantum state $|\psi\rangle = \sum_i c_i |\phi_i\rangle$. $|\psi\rangle$ is said to be coherent or to be in a linear superposition of the basis states $|\phi_i\rangle$, and in general the coefficients $c_i$ are complex. A postulate of quantum mechanics is that if a coherent system interacts in any way with its environment (by being measured, for example), the superposition is destroyed. This loss of coherence is governed by the wave function $\psi$. The coefficients $c_i$ are called probability amplitudes, and $|c_i|^2$ gives the probability of $|\psi\rangle$ being measured in the state $|\phi_i\rangle$ . Note that the wave function $\psi$ describes a real physical system that must collapse to exactly one basis state. Therefore, the probabilities governed by the amplitudes $c_i$ must sum to unity. A two-state quantum system is used as the basic unit of quantum computation. Such a system is referred to as a quantum bit or qubit and naming the two states $|0\rangle$ and $|1\rangle$, it is easy to see why this is so.

## 2.2 Operators

*Operators* on a Hilbert space describe how one wave function is changed into another and they may be represented as matrices acting on vectors (the notation $|\cdot\rangle$ indicates a column vector and the $\langle\cdot|$ a [complex conjugate] row vector). Using operators, an eigenvalue equation can be written $A |\phi_i\rangle = a_i |\phi_i\rangle$, where $a_i$ is the eigenvalue. The solutions $|\phi_i\rangle$ to such an equation are called eigenstates and can be used to construct the basis of a Hilbert space as discussed in Section 2.1. In the quantum formalism, all properties are represented as operators whose eigenstates are the basis for the Hilbert space associated with that property and whose eigenvalues are the quantum allowed values for that property. It is important to note that operators in quantum mechanics must be linear operators and further that they must be unitary.

## 2.3 Interference

*Interference* is a familiar wave phenomenon. Wave peaks that are in phase interfere constructively while those that are out of phase interfere destructively. This is a phenomenon common to all kinds of wave mechanics from water waves to optics. The well known double slit experiment demonstrates empirically that at the quantum level interference also applies to the probability waves of quantum mechanics. The wave function interferes with itself through the action of an operator – the different parts of the wave function interfere constructively or destructively according to their relative phases just like any other kind of wave.

## 2.4 Entanglement

Entanglement is the potential for quantum systems to exhibit correlations that cannot be accounted for classically. From a computational standpoint, entanglement seems intuitive enough – it is simply the fact that correlations can exist between different qubits – for example if one qubit is in the $|1\rangle$ state, another will be in the $|1\rangle$ state. However, from a physical standpoint, entanglement is little understood. The questions of what exactly it is and how

it works are still not resolved. What makes it so powerful (and so little understood) is the fact that since quantum states exist as superpositions, these correlations exist in superposition as well. When coherence is lost, the proper correlation is somehow communicated between the qubits, and it is this "communication" that is the crux of entanglement. Mathematically, entanglement may be described using the density matrix formalism. The density matrix $\rho_\psi$ of a quantum state $|\psi\rangle$ is defined as $\rho_\psi = |\psi\rangle\langle\psi|$ For example, the quantum

state $|\xi\rangle = \frac{1}{\sqrt{2}}|00\rangle + \frac{1}{\sqrt{2}}|01\rangle$ appears in vector form as $|\xi\rangle = \frac{1}{\sqrt{2}}\begin{pmatrix} 1 \\ 1 \\ 0 \\ 0 \end{pmatrix}$ and it may

also be represented as the density matrix $\rho_\xi = |\xi\rangle\langle\xi| = \frac{1}{2}\begin{pmatrix} 1 & 1 & 0 & 0 \\ 1 & 1 & 0 & 0 \\ 0 & 0 & 0 & 0 \\ 0 & 0 & 0 & 0 \end{pmatrix}$ while the

state $|\psi\rangle = \frac{1}{\sqrt{2}}|00\rangle + \frac{1}{\sqrt{2}}|11\rangle$ is represented as $\rho_\psi = |\psi\rangle\langle\psi| = \frac{1}{2}\begin{pmatrix} 1 & 0 & 0 & 1 \\ 0 & 0 & 0 & 0 \\ 0 & 0 & 0 & 0 \\ 1 & 0 & 0 & 1 \end{pmatrix}$

where the matrices and vectors are indexed by the state labels 00,..., 11. Notice that $\rho_\xi$ can be factorized as $\rho_\xi = \frac{1}{2}\left(\begin{pmatrix} 1 & 0 \\ 0 & 0 \end{pmatrix} \otimes \begin{pmatrix} 1 & 1 \\ 1 & 1 \end{pmatrix}\right)$ where $\otimes$ is the normal tensor product. On the other hand, $\rho_\psi$ can not be factorized. States that can not be factorized are said to be entangled, while those that can be factorized are not. There are different degrees of entanglement and much work has been done on better understanding and quantifying it [10, 11]. Finally, it should be mentioned that while interference is a quantum property that has a classical cousin, entanglement is a completely quantum phenomenon for which there is no classical analog. It has proven to be a powerful computational resource in some cases and a major hindrance in others.

To summarize, quantum computation can be defined as representing the problem to be solved in the language of quantum states and then producing operators that drive the system (via interference and entanglement) to a final state such that when the system is observed there is a high probability of finding a solution.

### 2.5 An Example – Quantum Search

One of the best known quantum algorithms searches an unordered database quadratically faster than any classical method [12, 13]. The algorithm begins with a superposition of all $N$ data items and depends upon an oracle that can recognize the target of the search. Classically, searching such a database requires $O(N)$ oracle calls; however, on a quantum computer, the task requires only $O(\sqrt{N})$ oracle calls. Each oracle call consists of a quantum operator that inverts the phase of the search target. An "inversion about average" operator then shifts amplitude towards the target state. After $\pi/4 * \sqrt{N}$ repetitions of this process, the system is measured and with high probability, the desired datum is the result.

## 3 A Simple Quantum Neural Network

We would like a QNN with features that make it easy for us to model, yet powerful enough to leverage quantum physics. We would like our QNN to:

- use known quantum algorithms and gates
- have weights which we can measure for each node

- work in classical simulations of reasonable size
- be able to transfer knowledge to classical systems

We propose a QNN that operates much like a classical ANN composed of several layers of perceptrons – an input layer, one or more hidden layers and an output layer. Each layer is fully connected to the previous layer. Each hidden layer computes a weighted sum of the outputs of the previous layer. If this is sum above a threshold, the node goes high, otherwise it stays low. The output layer does the same thing as the hidden layer(s), except that it also checks its accuracy against the target output of the network. The network as a whole computes a function by checking which output bit is high. There are no checks to make sure exactly one output is high. This allows the network to learn data sets which have one output high or binary-encoded outputs.

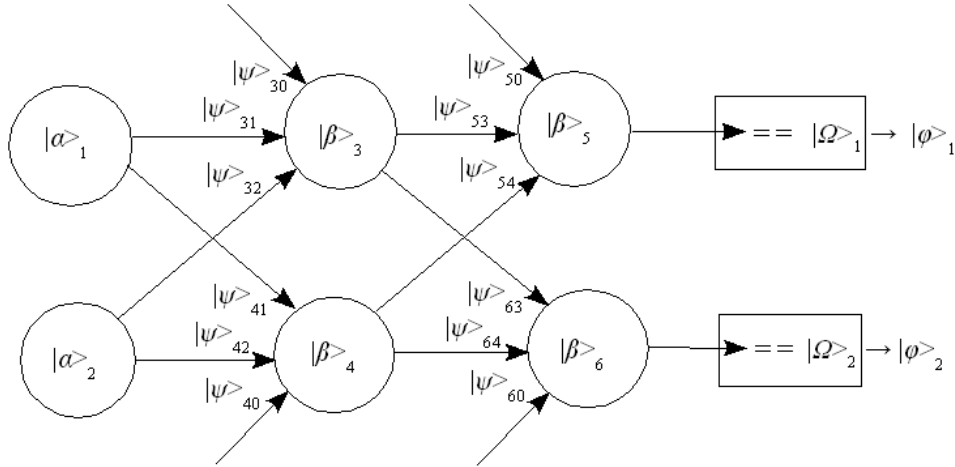

Figure 1: Simple QNN to compute XOR function

The QNN in Figure 1 is an example of such a network, with sufficient complexity to compute the XOR function. Each input node $i$ is represented by a register, $|\alpha\rangle_i$. The two hidden nodes compute a weighted sum of the inputs, $|\psi\rangle_{i1}$ and $|\psi\rangle_{i2}$, and compare the sum to a threshold weight, $|\psi\rangle_{i0}$. If the weighted sum is greater than the threshold the node goes high. The $|\beta\rangle_k$ represent internal calculations that take place at each node. The output layer works similarly, taking a weighted sum of the hidden nodes and checking against a threshold. The QNN then checks each computed output and compares it to the target output, $|\Omega\rangle_j$ sending $|\varphi\rangle_j$ high when they are equivalent. The performance of the network is denoted by $|\rho\rangle$, which is the number of computed outputs equivalent to their corresponding target output.

At the quantum gate level, the network will require $O(blm + m^2)$ gates for each node of the network. Here $b$ is the number of bits used for floating point arithmetic in $|\beta\rangle$, $l$ is the number of bits for each weight and $m$ is the number of inputs to the node [14]-[15].

The overall network works as follows on a training set. In our example, the network has two input parameters, so all $n$ training examples will have two input registers. These are represented as $|\alpha\rangle_{11}$ to $|\alpha\rangle_{n2}$. The target answers are kept in registers $|\Omega\rangle_{11}$ to $|\Omega\rangle_{n2}$. Each hidden or output node has a weight vector, represented by $|\psi\rangle_i$, each vector containing weights for each of its inputs. After classifying a training example, the registers $|\varphi\rangle_1$ and $|\varphi\rangle_2$ reflect the networks ability to classify that the training example. As a simple measure of performance, we increment $|\rho\rangle$ by the sum of all $|\varphi\rangle_i$. When all training examples have

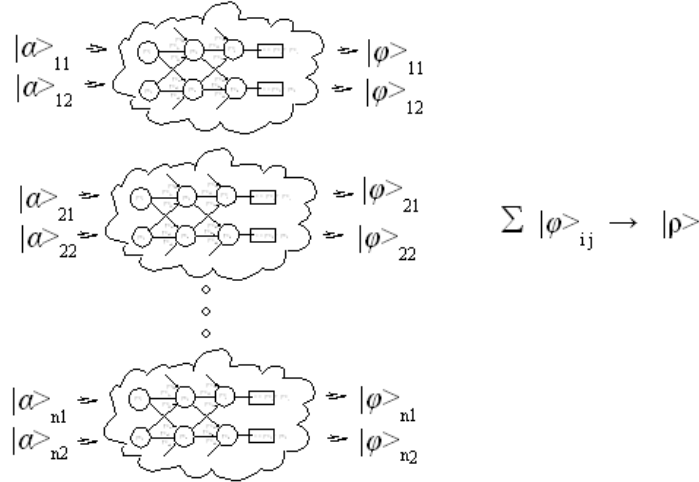

Figure 2: QNN Training

been classified, $|\rho\rangle$ will be the sum of the output nodes that have the correct answer throughout the training set and will range between zero and the number of training examples times the number of output nodes.

## 4    Using Quantum Search to Learn Network Weights

One possibility for training this kind of a network is to search through the possible weight vectors for one which is consistent with the training data. Quantum searches have been used already in quantum learning [16] and many of the problems associated with them have already been explored [17]. We would like to find a solution which classifies all training examples correctly; in other words we would like $|\rho\rangle = n * m$ where $n$ is the number of training examples and $m$ is the number of output nodes. Since we generally do not know how many weight vectors will do this, we use a generalization of the original search algorithm [18], intended for problems where the number of solutions $t$ is unknown. The basic idea is that we will put $|\psi\rangle$ into a superposition of all possible weight vectors and search for one which classifies all training examples correctly.

We start out with $|\psi\rangle$ as a superposition of all possible weight vectors. All other registers ($|\beta\rangle$, $|\varphi\rangle$, $|\rho\rangle$), besides the inputs and target outputs are initialized to the state $|0\rangle$. We then classify each training example, updating the performance register, $|\rho\rangle$. By using a superposition we classify the training examples with respect to every possible weight vector simultaneously. Each weight vector is now entangled with $|\rho\rangle$ in such a way that $|\rho\rangle$ corresponds with how well every weight vector classifies all the training data. In this case, the oracle for the quantum search is $|\rho\rangle = n * m$, which corresponds to searching for a weight vector which correctly classifies the entire set.

Unfortunately, searching the weight vectors while entangled with $|\rho\rangle$ would cause unwanted weight vectors to grow that would be entangled with the performance metric we are looking for. The solution is to disentangle $|\psi\rangle$ from the other registers after inverting the phase of those weights which match the search criteria, based on $|\rho\rangle$. To do this the entire network will need to be uncomputed, which will unentangle all the registers and set them back to their initial values. This means that the network will need to be recomputed

each time we make an oracle call and after each measurement.

There are at least two things about this algorithm that are undesirable. First, not all training data will have any solution networks that correctly classify all training instances. This means that nothing will be marked by the search oracle, so every weight vector will have an equal chance of being measured. It is also possible that even when a solution does exist, it is not desirable because it over fits the training data. Second, the amount of time needed to find a vector which correctly classifies the training set is $O(\sqrt{2^b/t})$, which has exponential complexity with respect to the number of bits in the weight vector.

One way to deal with the first problem is to search until we find a solution which covers an acceptable percentage, $p$, of the training data. In other words, the search oracle is modified to be $|\rho\rangle \geq n * m * p$. The second problem is addressed in the next section.

## 5    Piecewise Weight Learning

Our quantum search algorithm gives us a good polynomial speed-up to the exponential task of finding a solution to the QNN. This algorithm does not scale well, in fact it is exponential in the total number of weights in the network and the bits per weight. Therefore, we propose a randomized training algorithm which searches each node's weight vector independently.

The network starts off, once again, with training examples in $|\alpha\rangle$, the corresponding answers in $|\Omega\rangle$, and zeros in all the other registers. A node is randomly selected and its weight vector, $|\psi\rangle_i$, is put into superposition. All other weight vectors start with random classical initial weights. We then search for a weight vector for this node that causes the entire network to classify a certain percentage, $p$, of the training examples correctly. This is repeated, iteratively decreasing $p$, until a new weight vector is found. That weight is fixed classically and the process is repeated randomly for the other nodes.

Searching each node's weight vector separately is, in effect, a random search through the weight space where we select weight vectors which give a good level of performance for each node. Each node takes on weight vectors that tend to increase performance with some amount of randomness that helps keep it out of local minima. This search can be terminated when an acceptable level of performance has been reached.

There are a few improvements to the basic design which help speed convergence. First, to insure that hidden nodes find weight vectors that compute something useful, a small performance penalty is added to weight vectors which cause a hidden node to output the same value for all training examples. This helps select weight vectors which contain useful information for the output nodes. Since each output node's performance is independent of the performance or all output nodes, the algorithm only considers the accuracy of the output node being trained when training an output node.

## 6    Results

We first consider the canonical XOR problem. Each of the hidden and the output nodes are thresholded nodes with three weights, one for each input and one for the threshold. For each weight 2 bits are used. Quantum search did well on this problem, finding a solution in an average of 2.32 searches.

The randomized search algorithm also did well on the XOR problem. After an average of 58 weight updates, the algorithm was able to correctly classify the training data. Since this is a randomized algorithm both in the number of iterations of the search algorithm before measuring and in the order which nodes update their weight vectors, the standard deviation for this method was much higher, but still reasonable. In the randomized search algorithm,

an epoch refers to finding and fixing the weight of a single node.

We also tried the randomized search algorithm for a few real-world machine learning problems: lenses, Hayes-Roth and the iris datasets [19]. The lenses data set is a data set that tries to predict whether people will need soft contact lenses, hard contact lenses or no contacts. The iris dataset details features of three different classes of irises. The Hayes-Roth dataset classifies people into different classes depending several attributes.

| Data Set | # Weight Qubits | Epochs | Weight Updates | Output Accuracy | Training Accuracy | Backprop |
|---|---|---|---|---|---|---|
| Iris | 32 | 23,000 | 225 | 98.23% | 97.79% | 96% |
| Lenses | 42 | 22,500 | 145 | 98.35% | 100.0% | 92% |
| Hayes-Roth | 68 | $5 \times 10^6$ | 9,200 | 88.76% | 82.98% | 83% |

Table 1: Training Results

The lenses data set can be solved with a network that has three hidden nodes. After between a few hundred to a few thousand iterations it usually finds a solution. This may be because it has a hard time with 2 bit weights, or because it is searching for perfect accuracy. The number of times a weight was fixed and updated was only 225 for this data set. The iris data set was normalized so that each input had a value between zero and one. The randomized search algorithm found the correct target for 97.79% of the output nodes.

Our results for the Hayes-Roth problem were also quite good. We used four hidden nodes with two bit weights for the hidden nodes. We had to normalize the inputs to range from zero to one once again so the larger inputs would not dominate the weight vectors. The algorithm found the correct target for 88.86% of the output nodes correctly in about 5,000,000 epochs. Note that this does not mean that it classified 88.86% of the training examples correctly since we are checking each output node for accuracy on each training example. The algorithm actually classified 82.98% of the training set correctly, which compares well with backpropagation's 83% [20].

## 7   Conclusions and Future Work

This paper proposes a simple quantum neural network and a method of training it which works well in quantum systems. By using a quantum search we are able to use a well-known algorithm for quantum systems which has already been used for quantum learning. The algorithm is able to search for solutions that cover an arbitrary percentage of the training set. This could be very useful for problems which require a very accurate solution. The drawback is that it is an exponential algorithm, even with the significant quadratic speedup.

A randomized version avoids some of the exponential increases in complexity with problem size. This algorithm is exponential in the number of qubits of each node's weight vector instead of in the composite weight vector of the entire network. This means the complexity of the algorithm increases with the number of connections to a node and the precision of each individual weight, dramatically decreasing complexity for problems with large numbers of nodes. This could be a great improvement for larger problems. Preliminary results for both algorithms have been very positive.

There may be quantum methods which could be used to improve current gradient descent and other learning algorithms. It may also be possible to combine some of these with a quantum search. An example would be to use gradient descent to try and refine a composite weight vector found by quantum search. Conversely, a quantum search could start with the weight vector of a gradient descent search. This would allow the search to start with an

accurate weight vector and search locally for weight vectors which improve overall performance. Finally the two methods could be used simultaneously to try and take advantage of the benefits of each technique.

Other types of QNNs may be able to use a quantum search as well since the algorithm only requires a weight space which can be searched in superposition. In addition, more traditional gradient descent techniques might benefit from a quantum speed-up themselves.

## References

[1] Alexandr Ezhov and Dan Ventura. Quantum neural networks. In Ed. N. Kasabov, editor, *Future Directions for Intelligent Systems and Information Science*. Physica-Verlang, 2000.

[2] Ajit Narayanan and Tammy Menneer. Quantum artificial neural network architectures and components. In *Information Sciences*, volume 124 nos. 1-4, pages 231–255, 2000.

[3] M. V. Altaisky. Quantum neural network. Technical report, 2001. http://xxx.lanl.gov/quant-ph/0107012.

[4] E. C. Behrman, J. Niemel, J. E. Steck, and S. R. Skinner. A quantum dot neural network. In *Proceedings of the 4th Workshop on Physics of Computation*, pages 22–24. Boston, 1996.

[5] Fariel Shafee. Neural networks with c-not gated nodes. Technical report, 2002. http://xxx.lanl.gov/quant-ph/0202016.

[6] Yukari Fujita and Tetsuo Matsui. Quantum gauged neural network: U(1) gauge theory. Technical report, 2002. http://xxx.lanl.gov/cond-mat/0207023.

[7] S. Gupta and R. K. P. Zia. Quantum neural networks. In *Journal of Computer and System Sciences*, volume 63 No. 3, pages 355–383, 2001.

[8] E. C. Behrman, V. Chandrasheka, Z. Wank, C. K. Belur, J. E. Steck, and S. R. Skinner. A quantum neural network computes entanglement. Technical report, 2002. http://xxx.lanl.gov/quant-ph/0202131.

[9] Michael A. Nielsen and Isaac L. Chuang. Quantum computation and quantum information. Cambridge University Press, 2000.

[10] V. Vedral, M. B. Plenio, M. A. Rippin, and P. L. Knight. Quantifying entanglement. In *Physical Review Letters*, volume 78(12), pages 2275–2279, 1997.

[11] R. Jozsa. Entanglement and quantum computation. In S. Hugget, L. Mason, K.P. Tod, T. Tsou, and N.M.J. Woodhouse, editors, *The Geometric Universe*, pages 369–379. Oxford University Press, 1998.

[12] Lov K. Grover. A fast quantum mechanical algorithm for database search. In *Proceedings of the 28th ACM STOC*, pages 212–219, 1996.

[13] Lov K. Grover. Quantum mechanics helps in searching for a needle in a haystack. In *Physical Review Letters*, volume 78, pages 325–328, 1997.

[14] Peter Shor. Polynomial-time algorithms for prime factorization and discrete logarithms on a quantum computer. In *SIAM Journal of Computing*, volume 26 no. 5, pages 1484–1509, 1997.

[15] Vlatko Vedral, Adriano Barenco, and Artur Ekert. Quantum networks for elementary arithmetic operations. In *Physical Review A*, volume 54 no. 1, pages 147–153, 1996.

[16] Dan Ventura and Tony Martinez. Quantum associative memory. In *Information Sciences*, volume 124 nos. 1-4, pages 273–296, 2000.

[17] Alexandr Ezhov, A. Nifanova, and Dan Ventura. Distributed queries for quantum associative memory. In *Information Sciences*, volume 128 nos. 3-4, pages 271–293, 2000.

[18] Michel Boyer, Gilles Brassard, Peter Høyer, and Alain Tapp. Tight bounds on quantum searching. In *Proceedings of the Fourth Workshop on Physics and Computation*, pages 36–43, 1996.

[19] C.L. Blake and C.J. Merz. UCI repository of machine learning databases, 1998. http://www.ics.uci.edu/~mlearn/MLRepository.html.

[20] Frederick Zarndt. A comprehensive case study: An examination of machine learning and connectionist algorithms. Master's thesis, Brigham Young University, 1995.
